# Microscopic Equations in Rough Energy Landscape for Neural Networks

**K. Y. Michael Wong**
Department of Physics,
The Hong Kong University of Science and Technology,
Clear Water Bay, Kowloon, Hong Kong.
E-mail: phkywong@usthk.ust.hk

## Abstract

We consider the microscopic equations for learning problems in neural networks. The aligning fields of an example are obtained from the cavity fields, which are the fields if that example were absent in the learning process. In a rough energy landscape, we assume that the density of the local minima obey an exponential distribution, yielding macroscopic properties agreeing with the first step replica symmetry breaking solution. Iterating the microscopic equations provide a learning algorithm, which results in a higher stability than conventional algorithms.

## 1 INTRODUCTION

Most neural networks learn iteratively by gradient descent. As a result, closed expressions for the final network state after learning are rarely known. This precludes further analysis of their properties, and insights into the design of learning algorithms. To complicate the situation, metastable states (i.e. local minima) are often present in the energy landscape of the learning space so that, depending on the initial configuration, each one is likely to be the final state.

However, large neural networks are mean field systems since the examples and weights strongly interact with each other during the learning process. This means that when one example or weight is considered, the influence of the rest of the system can be regarded as a background satisfying some averaged properties. The situation is similar to a number of disordered systems such as spin glasses, in which mean field theories are applicable (Mézard, Parisi & Virasoro, 1987). This explains the success of statistical mechanical techniques such as the replica method in deriving the macroscopic properties of neural networks, e.g. the storage capacity (Gardner & Derrida 1988), generalization ability (Watkin, Rau & Biehl 1993). The replica

method, though, provides much less information on the microscopic conditions of the individual dynamical variables.

An alternative mean field approach is the cavity method. It is a generalization of the Thouless-Anderson-Palmer approach to spin glasses, which started from microscopic equations of the system elements (Thouless, Anderson & Palmer, 1977). Mézard applied the method to neural network learning (Mézard, 1989). Subsequent extensions were made to the teacher-student perceptron (Bouten, Schietse & Van den Broeck 1995), the AND machine (Griniasty, 1993) and the multiclass perceptron (Gerl & Krey, 1995). They yielded macroscopic properties identical to the replica approach, but the microscopic equations were not discussed, and the existence of local minima was neglected.

Recently, the cavity method was applied to general classes of single and multilayer networks with smooth energy landscapes, i.e. without the local minima (Wong, 1995a). The aligning fields of the examples satisfy a set of microscopic equations. Solving these equations iteratively provides a learning algoirthm, as confirmed by simulations in the maximally stable perceptron and the committee tree. The method is also useful in solving the dynamics of feedforward networks which were unsolvable previously (Wong, 1995b).

Despite its success, the theory is so far applicable only to the regime of smooth energy landscapes. Beyond this regime, a stability condition is violated, and local minima begin to appear (Wong, 1995a). In this paper I present a mean field theory for the regime of rough energy landscapes. The complete analysis will be published elsewhere and here I sketch the derivations, emphasizing the underlying physical picture. As shown below, a similar set of microscopic equations hold in this case, as confirmed by simulations in the committee tree. In fact, we find that the solutions to these equations have a higher stability than other conventional learning algorithms.

## 2   MICROSCOPIC EQUATIONS FOR SMOOTH ENERGY LANDSCAPES

We proceed by reviewing the cavity method for the case of smooth energy landscapes. For illustration we consider the single layer neural network (for two layer networks see Wong, 1995a). There are $N \gg 1$ input nodes $\{S_j\}$ connecting to a single output node by the synaptic weights $\{J_j\}$. The output state is determined by the sign of the local field at the output node, i.e. $S_{out} = \text{sgn}(\sum_j J_j S_j)$. Learning a set of $p$ examples means to find the weights $\{J_j\}$ such that the network gives the correct input-to-output mapping for the examples. If example $\mu$ maps the inputs $S_j^\mu$ to the output $O^\mu$, then a successful learning process should find a weight vector $J_j$ such that $\text{sgn}(\sum_j J_j \xi_j^\mu) = 1$, where $\xi_j^\mu = O^\mu S_j^\mu$. Thus the usual approach to learning is to first define an energy function (or error function) $E = \sum_\mu g(\Lambda_\mu)$, where $\Lambda_\mu \equiv \sum_j J_j \xi_j^\mu / \sqrt{N}$ are the aligning fields, i.e. the local fields in the direction of the correct output, normalized by the factor $\sqrt{N}$. For example, the Adatron algorithm uses the energy function $g(\Lambda) = (\kappa - \Lambda)\Theta(\kappa - \Lambda)$ where $\kappa$ is the stability parameter and $\Theta$ is the step function (Anlauf & Biehl, 1989). Next, one should minimize $E$ by gradient descent dynamics. To avoid ambiguity, the weights are normalized to $\sum_j S_j^2 = \sum_j J_j^2 = N$.

The cavity method uses a self-consistency argument to consider what happens when a set of $p$ examples is expanded to $p + 1$ examples. The central quantity in this method is the *cavity field*. For an added example labelled 0, the cavity field is the aligning field when it is fed to a network which learns examples 1 to $p$ (but

never learns example 0), i.e. $t_0 \equiv \sum_j J_j \xi_j^0 / \sqrt{N}$. Since the original network has no information about example 0, $J_j$ and $\xi_j^0$ are uncorrelated. Thus the cavity field obeys a Gaussian distribution for random example inputs.

After the network has learned examples 0 to $p$, the weights adjust from $\{J_j\}$ to $\{J_j^0\}$, and the cavity field $t_0$ adjusts to the generic aligning field $\Lambda_0$. As shown schematically in Fig. 1(a), we assume that the adjustments of the aligning fields of the original examples are small, typically of the order $O(N^{-1/2})$. Perturbative analysis concludes that *the aligning field is a well defined function of the cavity field,* i.e. $\Lambda_0 = \lambda(t_0)$ where $\lambda(t)$ is the inverse function of

$$t = \lambda + \gamma g'(\lambda), \tag{1}$$

and $\gamma$ is called the *local susceptibility.* The cavity fields satisfy a set of self-consistent equations

$$t_\mu = \sum_{\nu \neq \mu} [\lambda(t_\nu) - t_\nu] Q_{\nu\mu} + \alpha \chi \lambda(t_\mu) \tag{2}$$

where $Q_{\nu\mu} = \sum_j \xi_j^\nu \xi_j^\mu / N$. $\chi$ is called *nonlocal susceptibility,* and $\alpha \equiv p/N$. The weights $J_j$ are given by

$$J_j = (1 - \alpha\chi)^{-1} \frac{1}{\sqrt{N}} \sum_\mu [\lambda(t_\mu) - t_\mu] \xi_j^\mu. \tag{3}$$

Noting the Gaussian distribution of the cavity fields, the macroscopic properties of the neural network, such as the storage capacity, can be derived, and the results are identical to those obtained by the replica method (Gardner & Derrida 1988).

However, the real advantage of the cavity method lies in the microscopic information it provides. The above equations can be iterated sequentially, resulting in a general learning algorithm. Simulations confirm that the equations are satisfied in the single layer perceptron, and their generalized version holds in the committee tree at low loading (Wong, 1995a).

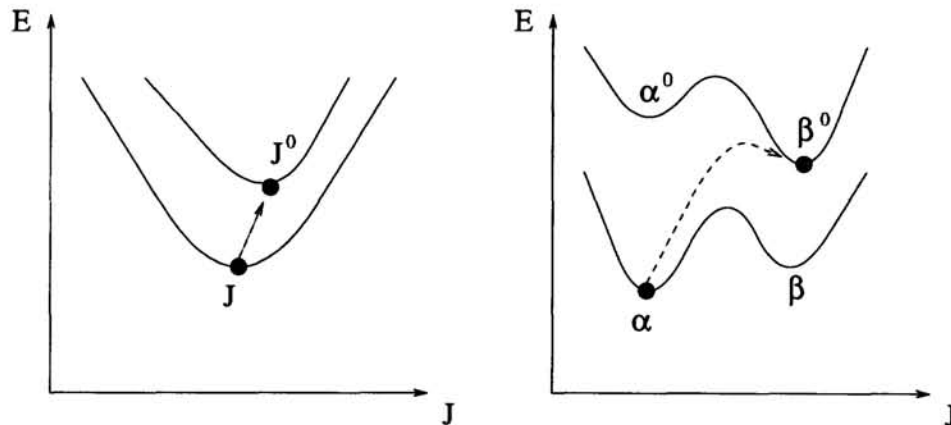

Figure 1: Schematic drawing of the change in the energy landscape in the weight space when example 0 is added, for the regime of (a) smooth energy landscape, (b) rough energy landscape.

## 3   MICROSCOPIC EQUATIONS FOR ROUGH ENERGY LANDSCAPES

However, the above argument holds under the assumption that the adjustment due to the addition of a new example is controllable. We can derive a stability condition for this assumption, and we find that it is equivalent to the Almeida-Thouless condition in the replica method (Mézard, Parisi & Virasoro, 1987).

An example for such instability occurs in the committee tree, which consists of hidden nodes $a = 1, \ldots, K$ with binary outputs, each fed by $K$ nonoverlapping groups of $N/K$ input nodes. The output of the committee tree is the majority state of the $K$ hidden nodes. The solution in the cavity method minimizes the change from the cavity fields $\{t_a\}$ to the aligning fields $\{\Lambda_a\}$, as measured by $\sum_a (\Lambda_a - t_a)^2$ in the space of correct outputs. Thus for a stability parameter $\kappa$, $\Lambda_a = \kappa$ when $t_a < \kappa$ and the value of $t_a$ is above median among the $K$ hidden nodes, otherwise $\Lambda_a = t_a$. Note that a discontinuity exists in the aligning field function. Now suppose $t_a < \kappa$ is the median, but the next highest value $t_b$ happens to be slightly less than $t_a$. Then the addition of example 0 may induce a change from $t_b < t_a$ to $t_{b0} > t_{a0}$. Hence $\Lambda_{b0}$ changes from $t_b$ to $\kappa$ whereas $\Lambda_{a0}$ changes from $\kappa$ to $t_{a0}$. The adjustment of the system is no longer small, and the previous perturbative analysis is not valid. In fact, it has been shown that all networks having a gap in the aligning field function are not stable against the addition of examples (Wong, 1995a).

To consider what happens beyond the stability regime, one has to take into account the rough energy landscape of the learning space. Suppose that the original global minimum for examples 1 to $p$ is $\alpha$. After adding example 0, a nonvanishing change to the system is induced, and the global minimum shifts to the neighborhood of the local minimum $\beta$, as schematically shown in Fig. 1(b). Hence the resultant aligning fields $\Lambda_0^\beta$ are no longer well-defined functions of the cavity fields $t_0^\alpha$. Instead they are well-defined functions of the cavity fields $t_0^\beta$. Nevertheless, one may expect that correlations exist between the states $\alpha$ and $\beta$.

Let $\sqrt{q_0}$ be the correlation between the network states, i.e. $\langle J_j^\alpha J_j^\beta \rangle = \sqrt{q_0}$. Since both states $\alpha$ and $\beta$ are determined in the absence of the added example 0, the correlation $\langle t_0^\alpha t_0^\beta \rangle = \sqrt{q_0}$ as well. Knowing that both $t_0^\alpha$ and $t_0^\beta$ obey Gaussian distributions, the cavity field distribution can be determined if we know the prior distribution of the local minima.

At this point we introduce the central assumption in the cavity method for rough energy landscapes: we assume that the number of local minima at energy $E$ obey an exponential distribution $d\aleph(E) = C \exp(-wE) dE$. Similar assumptions have been used in specifying the density of states in disordered systems (Mézard, Parisi & Virasoro 1987). Thus for single layer networks (and for two layer networks with appropriate generalizations), the cavity field ditribution is given by

$$P(t_0^\beta | t_0^\alpha) = \frac{G(t_0^\beta | t_0^\alpha) \exp[-w \Delta E(\lambda(t_0^\beta))]}{\int dt_0^\beta G(t_0^\beta | t_0^\alpha) \exp[-w \Delta E(\lambda(t_0^\beta))]}, \tag{4}$$

where $G(t_0^\beta | t_0^\alpha)$ is a Gaussian distribution. $w$ is a parameter describing the distribution, and $\lambda(t_0^\beta)$ is the aligning field function. The weights $J_j^\beta$ are given by

$$J_j^\beta = (1 - \alpha\chi)^{-1} \frac{1}{\sqrt{N}} \sum_\mu [\lambda(t_\mu^\beta) - t_\mu^\beta] \xi_j^\mu. \tag{5}$$

Noting the Gaussian distribution of the cavity fields, self-consistent equations for both $q_0$ and the local susceptibility $\gamma$ can be derived.

To determine the distribution of local minima, namely the parameters $C$ and $w$, we introduce a "free energy" $F(p, N)$ for $p$ examples and $N$ input nodes, given by $d\aleph(E) = \exp[w(F(p, N) - E)]dE$. This "free energy" determines the averaged energy of the local minima and should be an extensive quantity, i.e. it should scale as the system size. Cavity arguments enable us to find an expression $F(p+1, N) - F(p, N)$. Similarly, we may consider a cavity argument for the addition of one input node, expanding the network size from $N$ to $N + 1$. This yields an expression for $F(p, N+1) - F(p, N)$. Since $F$ is an extensive quantity, $F(p, N)$ should scale as $N$ for a given ratio $\alpha = p/N$. This implies

$$\frac{F}{N} = \alpha(F(p+1, N) - F(p, N)) + (F(p, N+1) - F(p, N)). \tag{6}$$

We have thus obtained an expression for the averaged energy of the local minima. Minimizing the free energy with respect to the parameter $w$ gives a self-consistent equation.

The three equations for $q_0$, $\gamma$ and $w$ completely determines the model. The macroscopic properties of the neural network, such as the storage capacity, can be derived, and the results are identical to the first step replica symmetry breaking solution in the replica method.

It remains to check whether the microscopic equations have been modified due to the roughening of the energy landscape. It turns out that while the cavity fields in the *initial* state $\alpha$ do not satisfy the microscopic equations (2), those at the *final* metastable state $\beta$ do, except that the nonlocal susceptibility $\chi$ has to be replaced by its average over the distribution of the local minima. In fact, the nonlocal susceptibility describes the reactive effects due to the background examples, which adjust on the addition of the new example. (Technically, this is called the Onsager reaction.) The adjustments due to hopping between valleys in a rough energy landscape have thus been taken into account.

## 4   SIMULATION RESULTS

To verify the theory, I simulate a committee tree learning random examples. Learning can be done by the more conventional Least Action algorithm (Nilsson 1965), or by iterating the microscopic equations.

We verify that the Least Action algorithm yields an aligning field function $\lambda(t)$ consistent with the cavity theory. Suppose the weights from input $j$ to hidden node $a$ is given by $J_{aj} = \sum_\mu x_{a\mu}\xi_j^\mu/\sqrt{N}$. Comparing with $J_{aj} = (1 - \alpha\chi)^{-1} \sum_\mu (\Lambda_{a\mu} - t_{a\mu})\xi_j^\mu/\sqrt{N}$, we estimate the nonlocal susceptibility $\chi$ by requiring the distribution of $\tilde{t}_{a\mu} \equiv \Lambda_{a\mu} - (1 - \alpha\chi)x_{a\mu}$ to have a zero first moment. $\tilde{t}_{a\mu}$ is then an estimate of $t_{a\mu}$. Fig. 2 shows the resultant relation between $\Lambda_{a\mu}$ and $\tilde{t}_{a\mu}$. It agrees with the predictions of the cavity theory. Fig. 3 shows the values of the stability parameter $\kappa$ measured from the Least Action algorithm and the microscopic equations. They have better agreement with the predictions of the rough energy landscape (first step replica symmetry breaking solution) rather than the smooth energy landscape (replica symmetric solution). Note that the microscopic equations yield a higher stability than the Least Action algorithm.

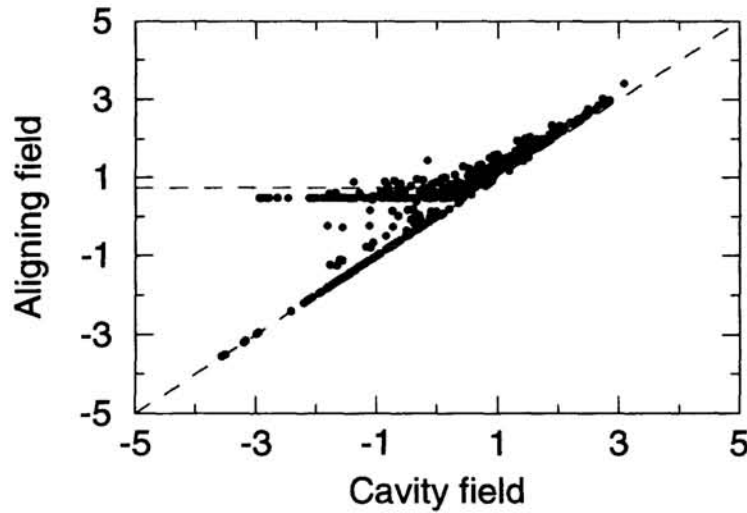

Figure 2: The aligning fields versus the cavity fields for a branch of the committee tree with $K = 3$, $\alpha = 0.8$ and $N = 600$. The dashed line is the prediction of the cavity theory for the regime of rough energy landscape.

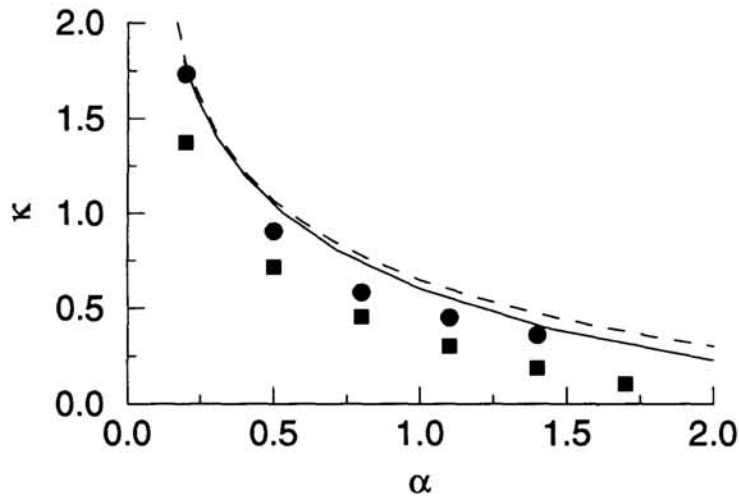

Figure 3: The stability parameter $\kappa$ versus the storage level $\alpha$ in the committee tree with $K = 3$ for the cavity theory of: (a) smooth energy landscape (dashed line), (b) rough energy landscape (solid line), and the simulation of: (c) iterating the microscopic equations (circles), (d) the Least Action algorithm (squares). Error bars are smaller than the size of the symbols.

## 5  CONCLUSION

In summary, we have derived the microscopic equations for neural network learning in the regime of rough energy landscapes. They turn out to have the same form as in the case of smooth energy landscape, except that the parameters are averaged over the distribution of local minima. Iterating the equations result in a learning algorithm, which yields a higher stability than more conventional algorithms in the committee tree. However, for high loading, the iterations may not converge.

The success of the present scheme lies its ability to take into account the underlying physical picture of many local minima of comparable energy. It correctly describes the experience that slightly different training sets may lead to vastly different neural networks. The stability parameter predicted by the rough landscape ansatz has a better agreement with simulations than the smooth one. It provides a physical interpretation of the replica symmetry breaking solution in the replica method. It is possible to generalize the theory to the physical picture with hierarchies of clusters of local minima, which corresponds to the infinite step replica symmetry breaking solution, though the mathematics is much more involved.

## Acknowledgements

This work is supported by the Hong Kong Telecom Institute of Information Techology, HKUST.

## References

Anlauf, J.K., & Biehl, M. (1989) The AdaTron: an adaptive perceptron algorithm. *Europhysics Letters* **10**(7):687-692.

Bouten, M., Schietse, J. & Van den Broeck, C. (1995) Gradient descent learning in perceptrons: A review of its possibilities. *Physical Review E* **52**(2):1958-1967.

Gardner, E. & Derrida, B. (1988) Optimal storage properties of neural network models. *Journal of Physics A: Mathematical and General* **21**(1):271-284.

Gerl, F. & Krey, U. (1995) A Kuhn-Tucker cavity method for generalization with applications to perceptrons with Ising and Potts neurons. *Journal of Physics A: Mathematical and General* **28**(23):6501-6516.

Griniasty, M. (1993) "Cavity-approach" analysis of the neural-network learning problem. *Physical Review E* **47**(6):4496-4513.

Mézard, M. (1989) The space of interactions in neural networks: Gardner's computation with the cavity method. *Journal of Physics A: Mathematical and General* **22**(12):2181-2190.

Mézard, M., Parisi, G. & Virasoro, M. (1987) *Spin Glass Theory and Beyond.* Singapore: World Scientific.

Nilsson, N.J. (1965) *Learning Machines.* New York: McGraw-Hill.

Thouless, D.J., Anderson, P.W. & Palmer, R.G. (1977) Solution of 'solvable model of a spin glass'. *Philosophical Magazine* **35**(3):593-601.

Watkin, T.L.H., Rau, A. & Biehl, M. (1993) The statistical mechanics of learning a rule. *Review of Modern Physics* **65**(2):499-556.

Wong, K.Y.M. (1995a) Microscopic equations and stability conditions in optimal neural networks. *Europhysics Letters* **30**(4):245-250.

Wong, K.Y.M. (1995b) The cavity method: Applications to learning and retrieval in neural networks. In J.-H. Oh, C. Kwon and S. Cho (eds.), *Neural Networks: The Statistical Mechanics Perspective*, pp. 175-190. Singapore: World Scientific.